# Dynamics of On-Line Gradient Descent Learning for Multilayer Neural Networks

**David Saad***
Dept. of Comp. Sci. & App. Math.
Aston University
Birmingham B4 7ET, UK

**Sara A. Solla†**
CONNECT, The Niels Bohr Institute
Blegdamsdvej 17
Copenhagen 2100, Denmark

## Abstract

We consider the problem of on-line gradient descent learning for general two-layer neural networks. An analytic solution is presented and used to investigate the role of the learning rate in controlling the evolution and convergence of the learning process.

Learning in layered neural networks refers to the modification of internal parameters $\{\mathbf{J}\}$ which specify the strength of the interneuron couplings, so as to bring the map $f_{\mathbf{J}}$ implemented by the network as close as possible to a desired map $\tilde{f}$. The degree of success is monitored through the *generalization error*, a measure of the dissimilarity between $f_{\mathbf{J}}$ and $\tilde{f}$.

Consider maps from an $N$-dimensional input space $\boldsymbol{\xi}$ onto a scalar $\zeta$, as arise in the formulation of classification and regression tasks. Two-layer networks with an arbitrary number of hidden units have been shown to be universal approximators [1] for such $N$-to-one dimensional maps. Information about the desired map $\tilde{f}$ is provided through independent examples $(\boldsymbol{\xi}^\mu, \zeta^\mu)$, with $\zeta^\mu = \tilde{f}(\boldsymbol{\xi}^\mu)$ for all $\mu$. The examples are used to train a student network with $N$ input units, $K$ hidden units, and a single linear output unit; the target map $\tilde{f}$ is defined through a teacher network of similar architecture except for the number $M$ of hidden units. We investigate the emergence of generalization ability in an *on-line* learning scenario [2], in which the couplings are modified after the presentation of each example so as to minimize the corresponding error. The resulting changes in $\{\mathbf{J}\}$ are described as a dynamical evolution; the number of examples plays the role of time.

In this paper we limit our discussion to the case of the soft-committee machine [2], in which all the hidden units are connected to the output unit with positive couplings of unit strength, and only the input-to-hidden couplings are adaptive.

*D.Saad@aston.ac.uk
†On leave from AT&T Bell Laboratories, Holmdel, NJ 07733, USA

Consider the student network: hidden unit $i$ receives information from input unit $r$ through the weight $J_{ir}$, and its activation under presentation of an input pattern $\boldsymbol{\xi} = (\xi_1, \ldots, \xi_N)$ is $x_i = \mathbf{J}_i \cdot \boldsymbol{\xi}$, with $\mathbf{J}_i = (J_{i1}, \ldots, J_{iN})$ defined as the vector of incoming weights onto the $i$-th hidden unit. The output of the student network is $\sigma(\mathbf{J}, \boldsymbol{\xi}) = \sum_{i=1}^{K} g(\mathbf{J}_i \cdot \boldsymbol{\xi})$, where $g$ is the activation function of the hidden units, taken here to be the error function $g(x) \equiv \mathrm{erf}(x/\sqrt{2})$, and $\mathbf{J} \equiv \{\mathbf{J}_i\}_{1 \leq i \leq K}$ is the set of input-to-hidden adaptive weights.

Training examples are of the form $(\boldsymbol{\xi}^\mu, \zeta^\mu)$. The components of the independently drawn input vectors $\boldsymbol{\xi}^\mu$ are uncorrelated random variables with zero mean and unit variance. The corresponding output $\zeta^\mu$ is given by a deterministic teacher whose internal structure is the same as for the student network but may differ in the number of hidden units. Hidden unit $n$ in the teacher network receives input information through the weight vector $\mathbf{B}_n = (B_{n1}, \ldots, B_{nN})$, and its activation under presentation of the input pattern $\boldsymbol{\xi}^\mu$ is $y_n^\mu = \mathbf{B}_n \cdot \boldsymbol{\xi}^\mu$. The corresponding output is $\zeta^\mu = \sum_{n=1}^{M} g(\mathbf{B}_n \cdot \boldsymbol{\xi}^\mu)$. We will use indices $i, j, k, l \ldots$ to refer to units in the student network, and $n, m, \ldots$ for units in the teacher network.

The error made by a student with weights $\mathbf{J}$ on a given input $\boldsymbol{\xi}$ is given by the quadratic deviation

$$\epsilon(\mathbf{J}, \boldsymbol{\xi}) \equiv \frac{1}{2} [\, \sigma(\mathbf{J}, \boldsymbol{\xi}) - \zeta \,]^2 = \frac{1}{2} \left[ \sum_{i=1}^{K} g(x_i) - \sum_{n=1}^{M} g(y_n) \right]^2 . \tag{1}$$

Performance on a typical input defines the generalization error $\epsilon_g(\mathbf{J}) \equiv \, < \epsilon(\mathbf{J}, \boldsymbol{\xi}) >_{\{\xi\}}$ through an average over all possible input vectors $\boldsymbol{\xi}$, to be performed implicitly through averages over the activations $\mathbf{x} = (x_1, \ldots, x_K)$ and $\mathbf{y} = (y_1, \ldots, y_M)$. Note that both $< x_i > = < y_n > = 0$; second order correlations are given by the overlaps among the weight vectors associated with the various hidden units: $< x_i \, x_k > = \mathbf{J}_i \cdot \mathbf{J}_k \equiv Q_{ik}$, $< x_i \, y_n > = \mathbf{J}_i \cdot \mathbf{B}_n \equiv R_{in}$, and $< y_n \, y_m > = \mathbf{B}_n \cdot \mathbf{B}_m \equiv T_{nm}$. Averages over $\mathbf{x}$ and $\mathbf{y}$ are performed using the resulting multivariate Gaussian probability distribution, and yield an expression for the generalization error in terms of the parameters $Q_{ik}$, $R_{in}$, and $T_{nm}$ [3]. For $g(x) \equiv \mathrm{erf}(x/\sqrt{2})$ the result is:

$$\epsilon_g(\mathbf{J}) = \frac{1}{\pi} \left\{ \sum_{ik} \arcsin \frac{Q_{ik}}{\sqrt{1 + Q_{ii}}\, \sqrt{1 + Q_{kk}}} + \sum_{nm} \arcsin \frac{T_{nm}}{\sqrt{1 + T_{nn}}\, \sqrt{1 + T_{mm}}} \right.$$
$$\left. - 2 \sum_{in} \arcsin \frac{R_{in}}{\sqrt{1 + Q_{ii}}\, \sqrt{1 + T_{nn}}} \right\} . \tag{2}$$

The parameters $T_{nm}$ are characteristic of the task to be learned and remain fixed. The overlaps $Q_{ik}$ and $R_{in}$, which characterize the correlations among the various student units and their degree of specialization towards the implementation of the desired task, are determined by the student weights $\mathbf{J}$ and evolve during training.

A gradient descent rule for the update of the student weights results in $\mathbf{J}_i^{\mu+1} = \mathbf{J}_i^\mu + \frac{\eta}{N} \delta_i^\mu \, \boldsymbol{\xi}^\mu$, where the learning rate $\eta$ has been scaled with the input size $N$, and

$$\delta_i^\mu \equiv g'(x_i^\mu) \left[ \sum_{n=1}^{M} g(y_n^\mu) - \sum_{j=1}^{K} g(x_j^\mu) \right] \tag{3}$$

is defined in terms of both the activation function $g$ and its derivative $g'$. The time evolution of the overlaps $R_{in}$ and $Q_{ik}$ can be explicitly written in terms of similar

difference equations. In the large $N$ limit the normalized number of examples $\alpha = \mu/N$ can be interpreted as a continuous time variable, leading to the equations of motion

$$\frac{dR_{in}}{d\alpha} = \eta < \delta_i\, y_n >_{\{\xi\}}\,,$$

$$\frac{dQ_{ik}}{d\alpha} = \eta < \delta_i\, x_k >_{\{\xi\}} + \eta < \delta_k\, x_i >_{\{\xi\}} + \eta^2 < \delta_i\, \delta_k >_{\{\xi\}}\,, \qquad (4)$$

to be averaged over all possible ways in which an example can be chosen at a given time step. The dependence on the current input $\xi$ is only through the activations $\mathbf{x}$ and $\mathbf{y}$; the corresponding averages can be performed analytically for $g(x) = \mathrm{erf}(x/\sqrt{2})$, resulting in a set of coupled first-order differential equations [3]. These dynamical equations are exact, and provide a novel tool used here to analyze the learning process for a general soft-committee machine with an arbitrary number $K$ of hidden units, trained to implement a task defined through a teacher of similar architecture except for the number $M$ of hidden units. In what follows we focus on uncorrelated teacher vectors of unit length, $T_{nm} = \delta_{nm}$.

The time evolution of the overlaps $R_{in}$ and $Q_{ik}$ follows from integrating the equations of motion (4) from initial conditions determined by a random initialization of the student vectors $\{\mathbf{J}_i\}_{1 \le i \le K}$. Random initial norms $Q_{ii}$ for the student vectors are taken here from a uniform distribution in the $[0, 0.5]$ interval. Overlaps $Q_{ik}$ between independently chosen student vectors $\mathbf{J}_i$ and $\mathbf{J}_k$, or $R_{in}$ between $\mathbf{J}_i$ and an unknown teacher vector $\mathbf{B}_n$ are small numbers, of order $1/\sqrt{N}$ for $N \gg K, M$, and taken here from a uniform distribution in the $[0, 10^{-12}]$ interval.

We show in Fig. 1a-c the evolution of the overlaps and generalization error for a *realizable* case: $K = M = 3$ and $\eta = 0.1$. This example illustrates the successive regimes of the learning process. The system quickly evolves into a symmetric subspace controlled by an unstable suboptimal solution which exhibits no differentiation among the various student hidden units. Trapping in the symmetric subspace prevents the specialization needed to achieve the optimal solution, and the generalization error remains finite, as shown by the plateau in Fig. 1c. The symmetric solution is unstable, and the perturbation introduced through the random initialization of the overlaps $R_{in}$ eventually takes over: the student units become specialized and the matrix $R$ of student-teacher overlaps tends towards the matrix $T$, except for a permutational symmetry associated with the arbitrary labeling of the student hidden units. The generalization error plateau is followed by a monotonic decrease towards zero once the specialization begins and the system evolves towards the optimal solution. The evolution of the overlaps and generalization error for the *unrealizable* case $K < M$ is characterized by qualitatively similar stages, except that the asymptotic behavior is controlled by a suboptimal solution which reflects the differences between student and teacher architectures.

Curves for the time evolution of the generalization error for different values of $\eta$ shown in Fig. 1d for $K = M = 3$ identify trapping in the symmetric subspace as a small $\eta$ phenomenon. We therefore consider the equations of motion (4) in the small $\eta$ regime. The term proportional to $\eta^2$ is neglected and the resulting truncated equations of motion are used to investigate a phase characterized by students of similar norms: $Q_{ii} = Q$ for all $1 \le i \le K$, similar correlations among themselves: $Q_{ik} = C$ for all $i \ne k$, and similar correlations with the teacher vectors: $R_{in} = R$ for all $1 \le i \le K$, $1 \le n \le M$. The resulting dynamical equations exhibit a fixed point solution at

$$Q^* = C^* = \frac{M}{K^2}\frac{M - K^2 + \sqrt{K^4 - K^2 + M^2}}{2M - 1} \quad \text{and} \quad R^* = \sqrt{\frac{Q^*}{M}} \qquad (5)$$

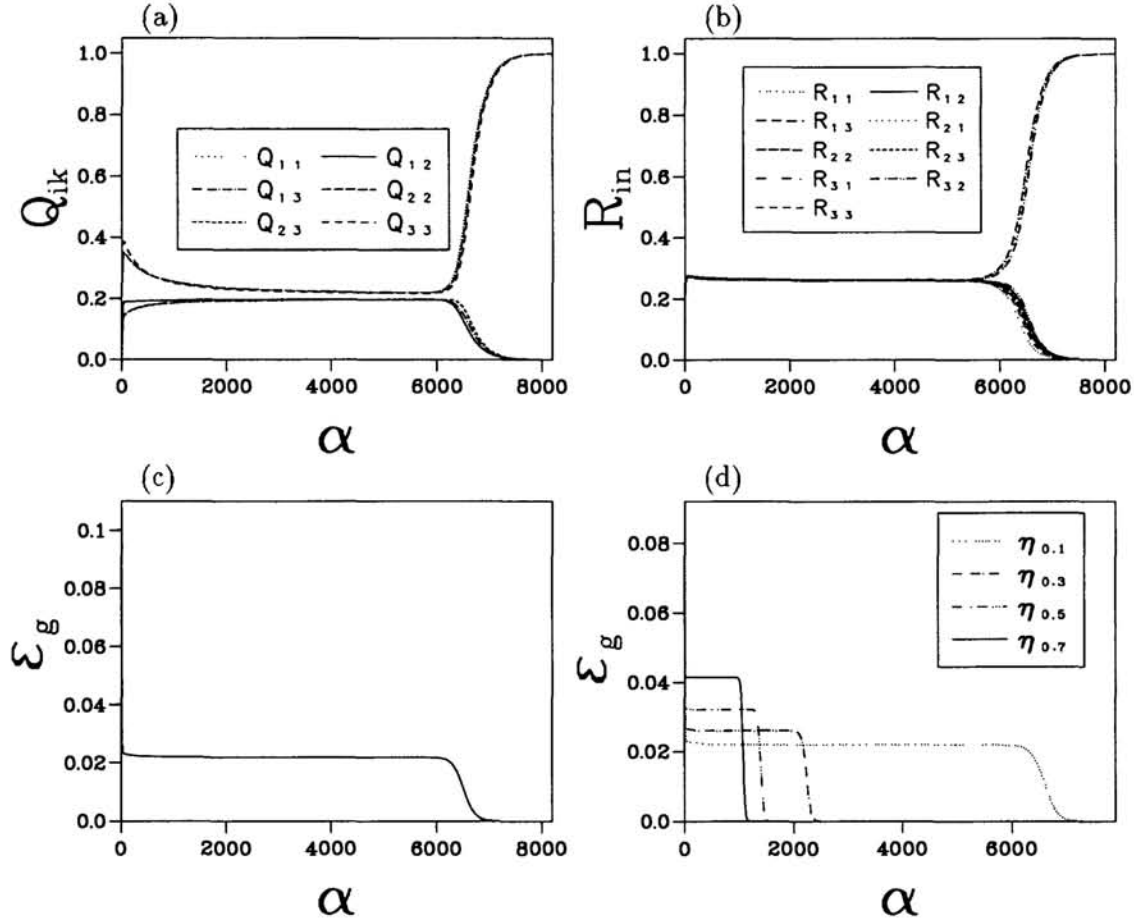

Figure 1: Dependence of the overlaps and the generalization error on the normalized number of examples $\alpha$ for a three-node student learning a three-node teacher characterized by $T_{nm} = \delta_{nm}$. Results for $\eta = 0.1$ are shown for (a) student-student overlaps $Q_{ik}$ and (b) student-teacher overlaps $R_{in}$. The generalization error is shown in (c), and again in (d) for different values of the learning rate.

for the general case, which reduces to

$$Q^* = C^* = \frac{1}{2K-1} \quad \text{and} \quad R^* = \sqrt{\frac{Q^*}{K}} = \frac{1}{\sqrt{K(2K-1)}} \tag{6}$$

in the realizable case $(K = M)$, where the corresponding generalization error is given by

$$\epsilon_g^* = \frac{K}{\pi}\left\{\frac{\pi}{6} - K\arcsin\left(\frac{1}{2K}\right)\right\} . \tag{7}$$

A simple geometrical picture explains the relation $Q^* = C^* = M(R^*)^2$ at the symmetric fixed point. The learning process confines the student vectors $\{\mathbf{J}_i\}$ to the subspace $S_B$ spanned by the set of teacher vectors $\{\mathbf{B}_n\}$. For $T_{nm} = \delta_{nm}$ the teacher vectors form an orthonormal set: $\mathbf{B}_n = \mathbf{e}_n$, with $\mathbf{e}_n \cdot \mathbf{e}_m = \delta_{nm}$ for $1 \leq n, m \leq M$, and provide an expansion for the weight vectors of the trained student: $\mathbf{J}_i^* = \sum_n R_{in}\mathbf{e}_n$. The student-teacher overlaps $R_{in}$ are independent of $i$ in the symmetric phase and independent of $n$ for an isotropic teacher: $R_{in} = R^*$ for all $1 \leq i \leq K$ and $1 \leq n \leq M$. The expansion $\mathbf{J}_i^* = R^* \sum_n \mathbf{e}_n$ for all $i$ results in $Q^* = C^* = M(R^*)^2$.

The length of the symmetric plateau is controlled by the degree of asymmetry in the initial conditions [2] and by the learning rate $\eta$. The small $\eta$ analysis predicts trapping times inversely proportional to $\eta$, in quantitative agreement with the shrinking plateau of Fig. 1d. The increase in the height of the plateau with decreasing $\eta$ is a second order effect, as the truncated equations of motion predict a unique value: $\epsilon_g^* = 0.0203$ for $K = M = 3$. The mechanism for the second order effect is revealed by an examination of Fig. 1a: the student-student overlaps do agree with the prediction $C^* = 0.2$ of the small $\eta$ analysis for $K = M = 3$, but the norms of the student vectors remain larger, at $Q = Q^* + \Delta$. The gap $\Delta$ between diagonal and off-diagonal elements is observed numerically to increase with increasing $\eta$, and is responsible for the excess generalization error. A first order expansion in $\Delta$ at $R = R^*$, $C = C^*$, and $Q = Q^* + \Delta$ yields

$$\epsilon_g = \frac{K}{\pi} \left\{ \frac{\pi}{6} - K \arcsin\left(\frac{1}{2K}\right) + \sqrt{\frac{2K-1}{2K+1}}\, \Delta \right\} \,, \qquad (8)$$

in agreement with the trend observed in Fig. 1d for the realizable case.

The excess norm $\Delta$ of the student vectors corresponds to a residual component in $\mathbf{J}_i$ not confined to the subspace $\mathcal{S}_B$. The weight vectors of the trained student can be written as $\mathbf{J}_i = R^* \sum_n \mathbf{e}_n + \mathbf{J}_i^\perp$, with $\mathbf{J}_i^\perp \cdot \mathbf{e}_n = 0$ for all $1 \leq n \leq M$. Student weight vectors are not constrained to be identical; they differ through orthogonal components $\mathbf{J}_i^\perp$ which are typically uncorrelated: $\mathbf{J}_i^\perp \cdot \mathbf{J}_k^\perp = 0$ for $i \neq k$. Correlations $Q_{ik} = C$ do satisfy $C = C^* = M(R^*)^2$, but norms $Q_{ii} = Q$ are given by $Q = Q^* + \Delta$, with $\Delta = \| \mathbf{J}^\perp \|^2$. Learning at very small $\eta$ tends to eliminate $\mathbf{J}^\perp$ and confine the student vectors to $\mathcal{S}_B$.

Escape from the symmetric subspace signals the onset of hidden unit specialization. As shown in Fig. 1b, the process is driven by a breaking of the uniformity of the student-teacher correlations: each student node becomes increasingly specialized to a specific teacher node, while its overlap with the remaining teacher nodes decreases and eventually decays to its asymptotic value. In the realizable case this asymptotic value is zero, while in the unrealizable case two different non-zero asymptotic values distinguish weak overlaps with teacher nodes imitated by other student vectors from more significant overlaps with those teacher nodes not specifically imitated by any of the student vectors.

The matrix of student-teacher overlaps can no longer be characterized by a unique parameter, as we need to distinguish between a dominant overlap $R$ between a given student node and the teacher node it begins to imitate, secondary overlaps $S$ between the same student node and the teacher nodes to which other student nodes are being assigned, and residual overlaps $U$ with the remaining teacher nodes. The student hidden nodes can be relabeled so as to bring the matrix of student-teacher overlaps to the form $R_{in} = R\delta_{in} + S(1-\delta_{in})\Theta(K-n) + U(1-\Theta(K-n))$, where the step function $\Theta$ is 0 for negative arguments and 1 otherwise. The emerging differentiation among student vectors results in a decrease of the overlaps $Q_{ik} = C$ for $i \neq k$, while their norms $Q_{ii} = Q$ increase. The matrix of student-student overlaps takes the form $Q_{ik} = Q\delta_{ik} + C(1-\delta_{ik})$.

Here we limit our description of the onset of specialization to the realizable case, for which $R_{in} = R\delta_{in} + S(1-\delta_{in})$. The small $\eta$ analysis is extended to allow for $S \neq R$ in order to describe the escape from the symmetric subspace. The resulting dynamical equations are linearized around the fixed point solution at $Q^* = C^* = 1/(2K-1)$ and $R^* = S^* = 1/\sqrt{K(2K-1)}$, and the generalization error is expanded around its fixed point value (7) to first order in the corresponding deviations $q$, $c$, $r$, and $s$. The analysis identifies a relevant perturbation with $q = c = 0$ and $s = -r/(K-1)$, which

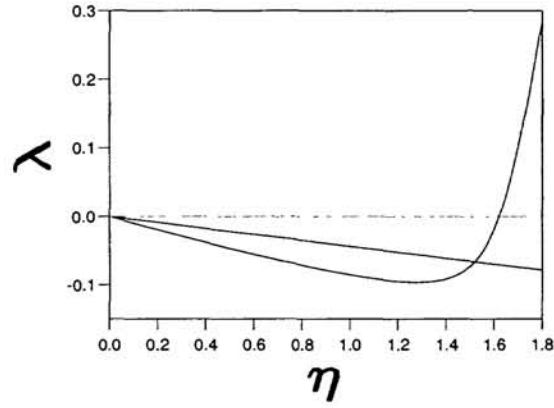

Figure 2: Dependence of the two leading decay eigenvalues on the learning rate $\eta$ in the realizable case: $\lambda_1$ (curved line) and $\lambda_2$ (straight line) are shown for $M = K = 3$.

leaves the generalization error unchanged and explains the behavior illustrated in Fig. 1a-b. It is the differentiation between $R$ and $S$ which signals the escape from the symmetric subspace; the differentiation between $Q$ and $C$ occurs for larger values of $\alpha$. The relevant perturbation corresponds to an enhancement of the overlap $R = R^* + r$ between a given student node and the teacher node it is learning to imitate, while the overlap $S = S^* + s$ between the same student node and the remaining teacher nodes is weakened. The time constant associated with this mode is $\tau = (\pi/2K)(2K - 1)^{1/2}(2K + 1)^{3/2}$, with $\tau \sim 2\pi K$ in the large $K$ limit.

It is in the subsequent convergence to an asymptotic solution that the realizable and unrealizable cases exhibit fundamental differences. We examine first the realizable scenario, in which the system converges to an optimal solution with perfect generalization.

As the specialization continues, the dominant overlaps $R$ grow, and the secondary overlaps $S$ decay to zero. Further specialization involves the decay to zero of the student-student correlations $C$ and the growth of the norms $Q$ of the student vectors. To investigate the convergence to the optimal solution we linearize the equations of motion around the asymptotic fixed point at $S^* = C^* = 0$, $R^* = Q^* = 1$, with $\epsilon_g^* = 0$. We describe convergence to the optimal solution by applying the full equations of motion (4) to a phase characterized by $R_{in} = R\delta_{in} + S(1 - \delta_{in})$ and $Q_{ik} = Q\delta_{ik} + C(1 - \delta_{in})$.

Linearization of the full equations of motion around the asymptotic fixed point results in four eigenvalues; the dependence of the two largest eigenvalues on $\eta$ is shown in Fig. 2 for $M = K = 3$. An initially slow mode corresponds to the eigenvalue $\lambda_2$, which remains negative for all values of $\eta$, while the eigenvalue $\lambda_1$ for the initially fast mode becomes positive as $\eta$ exceeds $\eta_{max}$, given by

$$\eta_{max} = \frac{\pi}{K} \frac{75 - 42\sqrt{3}}{25\sqrt{3} - 42} \tag{9}$$

to first order in $1/K$. The optimal solution with $\epsilon_g^* = 0$ is not accessible for $\eta > \eta_{max}$. Exponential convergence of $R$, $S$, $C$, and $Q$ to their optimal values is guaranteed for all learning rates in the range $(0, \eta_{max})$; in this regime the generalization error decays exponentially to $\epsilon_g^* = 0$, with a rate controlled by the slowest decay mode. An expansion of $\epsilon_g$ in terms of $r = 1 - R$, $s$, $c$, and $q = 1 - Q$ reveals that of the leading modes whose eigenvalues are shown in Fig. 2 only the mode associated with $\lambda_1$ contributes to the decay of the linear term, while the decay of the second order term is controlled by the mode associated with $\lambda_2$ and dominates the convergence if $2\lambda_2 < \lambda_1$. The learning rate $\eta_{opt}$ which guarantees fastest asymptotic decay for the generalization error follows from $\lambda_1(\eta_{opt}) = 2\lambda_2(\eta_{opt})$.

The asymptotic convergence of unrealizable learning is an intrinsically more complicated process that cannot be described in closed analytic form. The asymptotic

values of the order parameters and the generalization error depend on the learning rate $\eta$; convergence to an optimal solution with minimal generalization error requires $\eta \to 0$ as $\alpha \to \infty$. Optimal values for the order parameters follow from a small $\eta$ analysis, equivalent to neglecting $\mathbf{J}^\perp$ and assuming student vectors confined to $\mathcal{S}_B$. The resulting expansion $\mathbf{J}_i = \sum_{n=1}^M R_{in}\mathbf{e}_n$, with $R_{ii} = R$, $R_{in} = S$ for $1 \le n \le K$, $n \ne i$, and $R_{in} = U$ for $K+1 \le n \le M$, leads to

$$Q = R^2 + (K-1)S^2 + (M-K)U^2 \ , \quad C = 2RS + (K-2)S^2 = (M-K)U^2 \ . \quad (10)$$

The equations of motion for the remaining parameters $R$, $S$, and $U$ exhibit a fixed point solution which controls the asymptotic behavior. This solution cannot be obtained analytically, but numerical results are well approximated to order $(1/K^3)$ by

$$R^* = 1 - \frac{6\sqrt{3}-3}{8}\frac{L}{K^2}\left(1-\frac{1}{K}\right) \ ,$$

$$S^* = \left(1-\frac{\sqrt{3}}{6}\right)\frac{L}{K^3} \ , \quad U^* = \frac{1}{K}\left(1-\frac{1}{2K^2}\right) \ , \quad (11)$$

where $L \equiv M - K$. The corresponding fixed point values $Q^*$ and $C^*$ follow from Eq. (10). Note that $R^*$ is lower than for the realizable case, and that correlations $U^*$ (significant) and $S^*$ (weaker) between student vectors and the teacher vectors they do not imitate are not eliminated. The asymptotic generalization error is given by

$$\epsilon_g^* = \frac{1}{24\pi}\frac{L}{K^2}\left[4K^2(\pi-3) + 4K(2\sqrt{3}-3) + 1\right] \quad (12)$$

to order $(1/K^2)$. Note its proportionality to the mismatch $L$ between teacher and student architectures.

Learning at fixed and sufficiently small $\eta$ results in exponential convergence to an asymptotic solution whose fixed point coordinates are shifted from the values discussed above. The solution is suboptimal; the resulting increase in $\epsilon_g^*$ from its optimal value (12) is easily obtained to first order in $\eta$, and it is also proportional to $L$. We have investigated convergence to the optimal solution (12) for schedules of the form $\eta(\alpha) = \eta_0/(\alpha-\alpha_0)^z$ for the decay of the learning rate. A constant rate $\eta_0$ is used for $\alpha \le \alpha_0$; the monotonic decrease of $\eta$ for $\alpha > \alpha_0$ is switched on after specialization begins. Asymptotic convergence requires $0 < z \le 1$; fastest decay of the generalization error is achieved for $z = 1/2$.

Specialization as described here and illustrated in Fig.1 is a simultaneous process in which each student node acquires a strong correlation with a specific teacher node while correlations to other teacher nodes decrease. Such synchronous escape from the symmetric phase is characteristic of learning scenarios where the target task is defined through an isotropic teacher. In the case of a graded teacher we find that specialization occurs through a sequence of escapes from the symmetric subspace, ordered according to the relevance of the corresponding teacher nodes [3].

**Acknowledgement** The work was supported by the EU grant CHRX-CT92-0063.

# References

[1] G. Cybenko, *Math. Control Signals and Systems* **2**, 303 (1989).

[2] M. Biehl and H. Schwarze, *J. Phys. A* **28**, 643 (1995).

[3] D. Saad and S. A. Solla, *Phys. Rev. E*, **52**, 4225 (1995).